# Circuits for VLSI Implementation of Temporally-Asymmetric Hebbian Learning

**Adria Bofill**        **Alan F. Murray**

**Damon P. Thompson**

Dept. of Electrical Engineering
The University of Edinburgh
Edinburgh, EH93JL, UK
*adria.bofill@ee.ed.ac.uk*
*alan.murray@ee.ed.ac.uk*
*damon.thompson@ee.ed.ac.uk*

## Abstract

Experimental data has shown that synaptic strength modification in some types of biological neurons depends upon precise spike timing differences between presynaptic and postsynaptic spikes. Several temporally-asymmetric Hebbian learning rules motivated by this data have been proposed. We argue that such learning rules are suitable to analog VLSI implementation. We describe an easily tunable circuit to modify the weight of a silicon spiking neuron according to those learning rules. Test results from the fabrication of the circuit using a $0.6\mu$m CMOS process are given.

## 1  Introduction

Hebbian learning rules modify weights of synapses according to correlations between activity at the input and the output of neurons. Most artificial neural networks using Hebbian learning are based on pulse-rate correlations between continuous-valued signals; they reduce the neural spike trains to mean firing rates and thus precise timing does not carry information. With this approach the spiking nature of biological neurons is just an efficient solution that evolution has produced to transmit analog information over an unreliable medium.

In recent years, recorded data have indicated that synaptic strength modifications are also induced by timing differences between pairs of presynaptic and postsynaptic spikes [1][2]. A class of learning rules derived from these experimental data is illustrated in Figure 1 [2]-[4]. The "causal/non-causal" basis of these Hebbian learning algorithms is present in all variants of this spike-timing dependent weight modification rule. When the presynaptic spike arrives at the synapse a few milliseconds

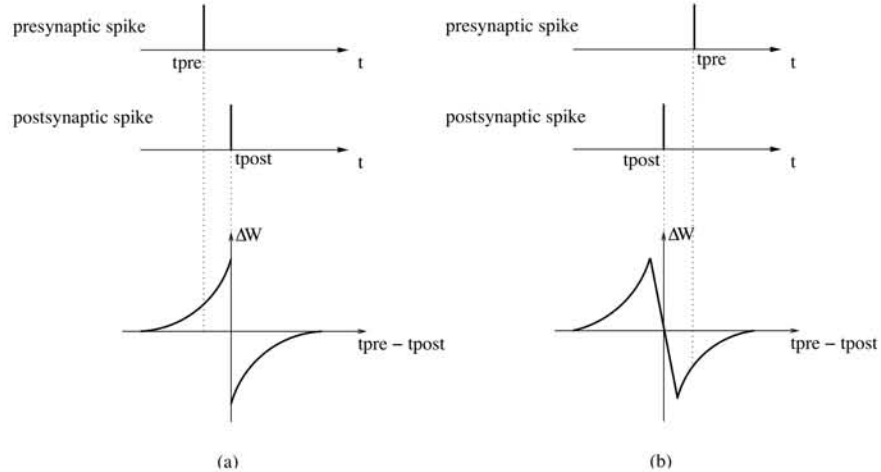

Figure 1: **Two temporally-asymmetric Hebbian learning rules drawing on experimental data.** The curves show the shape of the weight change ($\Delta$W) for differences between the firing times of the presynaptic ($t_{pre}$) and the postsynaptic ($t_{post}$) neurons. When the presynaptic spike arrives at the synapse a few ms before the postsynaptic neuron fires, the weight of the synapse is increased. If the postsynaptic neuron fires first, the weight is decreased.

before an output spike is generated, the synaptic efficiency increases. In contrast, when the postsynaptic neuron fires first, the efficiency of the synapse is weakened. Hence, only those synapses that receive spikes that appear to contribute to the generation of the postsynaptic spike are reinforced. In [5] a similar spike-timing difference based learning rule has been used to learn input sequence prediction in a recurrent network. Studies reported in [4] indicate that the positive (potentiation) element of the learning curve must be smaller than the negative (depression) to obtain stable competitive weight modification.

Pulse signal representation has been used extensively in hardware implementations of artificial neural networks [6][7]. Such systems use pulses as a mere technological solution to benefit from the robustness of binary signal transmission while making use of analog circuitry for the elementary computation units. However, they do not exploit the relative timing differences between individual pulses to compute. Also, analog hardware is not well-suited to the complexity of most artificial neural network algorithms. The learning rules presented in Figure 1 are suitable for analog VLSI because: (a) the signals involved in the weight modification are local to the neuron, (b) no temporal averaging of the presynaptic or postsynaptic activity is needed and (c) they are remarkably simple compared to complex neural algorithms that impose mathematical constraints in terms of accuracy and precision. An analog VLSI implementation of a similar, but more complex, spike-timing dependent learning rule can be found in [8].

We describe a circuit that implements the spike-timing dependent weight change described above along with the test results from a fabricated chip. We have focused on the implementation of the weight modification circuits, as VLSI spiking neurons with tunable membrane time constant and refractory period have already been proposed in [9] and [10].

## 2    Learning circuit description

Figure 2 shows the weight change circuit and Figure 3 the form of signals required to drive learning. These driving signals are generated by the circuits described in Figure 4. The voltage across the weight capacitor, $C_w$ in Figure 2, is modified according to the spike-timing dependent weight change rule discussed above. The weight change, $\Delta W$, is defined as $-\Delta V_w$ so that the leakage of the capacitor leads $V_w$ in the direction of weight decay. The circuits presented allow the control of: (a) the abruptness of the transition between potentiation and depression at the origin, (b) the difference between the areas under the curve in the potentiation and depression regions, (c) the absolute value of the area under each side of the curve and (d) the time constant of the curve decay.

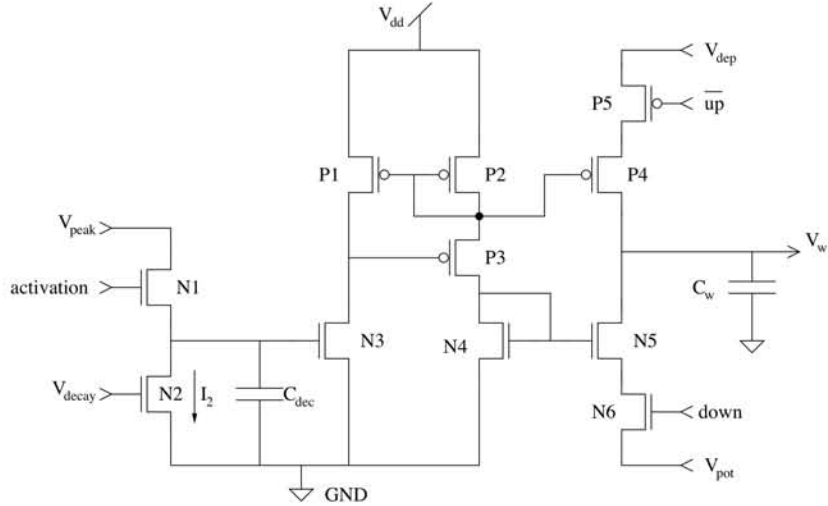

Figure 2: **Weight change circuit**

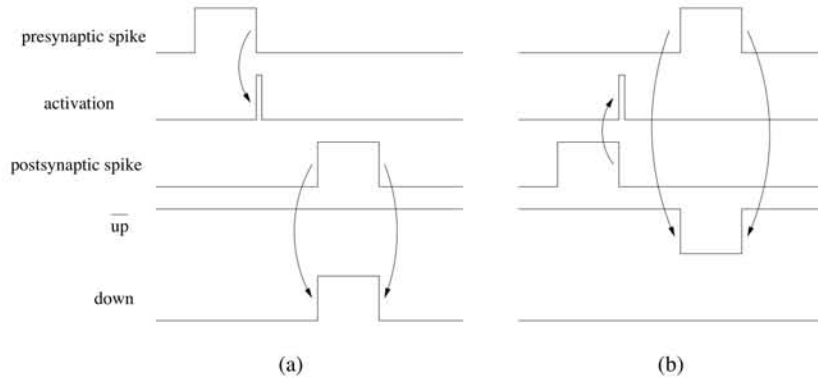

Figure 3: **Stimulus for the weight change circuit**

The weight change circuit of Figure 2 works as follows. When a falling edge of either a postsynaptic or a presynaptic spike occurs, a short *activation* pulse is generated which causes $C_{dec}$ to be charged to $V_{peak}$ through transistor N1. The charge accumulated in $C_{dec}$ will leak to ground with a rate set by $V_{decay}$. The

resulting voltage at the gate of N3 produces a current flowing through P2-P3-N4. If a presynaptic spike is active after the falling edge of a postsynaptic spike an active-low $\overline{up}$ pulse is applied to the gate of transistor P5. Thus, the current flowing through N3 is mirrored to transistor P4 causing an increase in the voltage across $C_w$ that corresponds to a decrease in the weight. In contrast, when a presynaptic spike precedes a postsynaptic spike an active-high *down* pulse is generated and the current in N3 is mirrored to N5-N6 resulting in a discharge of $C_w$.

As the current in N2 is constant, the current integrated by $C_w$ displays an exponential decay, if $V_{peak}$ is such that N3 is in sub-threshold mode. Hence, the rate of decay of the learning curve is fixed by the ratio $I_2/C_{dec}$. The abruptness of the transition zone between potentiation and depression is set by the duration of both the presynaptic and postsynaptic spike. Finally, an imbalance between the areas under the positive and negative side of the curve can be introduced via $V_{dep}$ and $V_{pot}$. The effect of all these circuit parameters is exemplified by the test results shown in the following section.

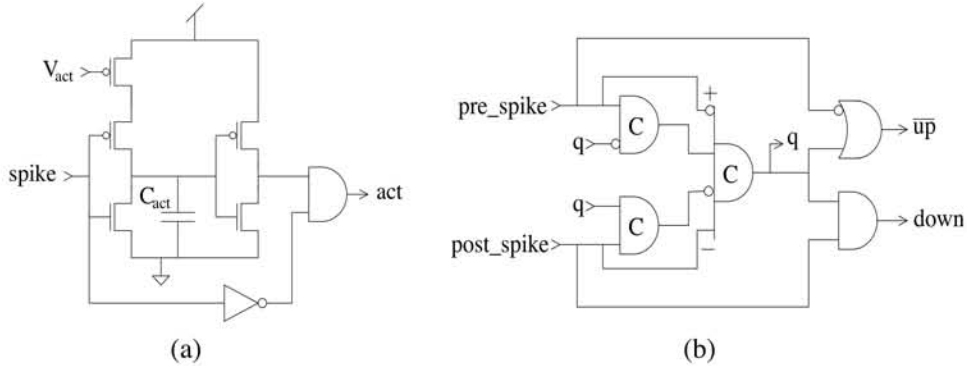

(a)                                                    (b)

Figure 4: **Learning drivers. (a) Delayed** *act* **pulse generator. (b) Asynchronous controller for** $\overline{up}$ **and** *down* **signals**

The circuit of Figure 4(a), present in both the presynaptic and postsynaptic neurons, generates a short *act* pulse with the falling edge of the output spike. The *act* pulses are *OR*ed at each synapse to produce the *activation* pulse applied to the weight change circuit of Figure 2.

The other two driving signals, $\overline{up}$ and *down*, are produced by a small asynchronous controller using standard and asymmetric C-elements [11] shown in Figure 4(b). The internal signal $q$ indicates if the last falling edge to occur corresponds to a pre ($q = 1$) or a postsynaptic spike ($q = 0$). This ensures that an $\overline{up}$ signal that decreases the weight is only generated when a presynaptic spike is active after the falling edge of a postsynaptic spike. Similarly *down* is activated only when the postsynaptic spike is active following a presynaptic spike falling edge.

Using the current flowing through N3 (Figure 2) to both increase and decrease the weight allows us to match the curve at the potentiation and depression regions at the expense of having to introduce the driving circuits of Figure 4.

## 3   Results from the temporally-asymmetric Hebbian chip

The circuit in Figure 2 has been fabricated in a $0.6\mu$m standard CMOS process. The driving signals (*down*, $\overline{up}$ and *activation*) are currently generated off-chip.

The circuit can be operated in the $\mu$s timescale, however, here we only present test results with time constants similar to those suggested by experimental data and studied using software models in [3]-[5].

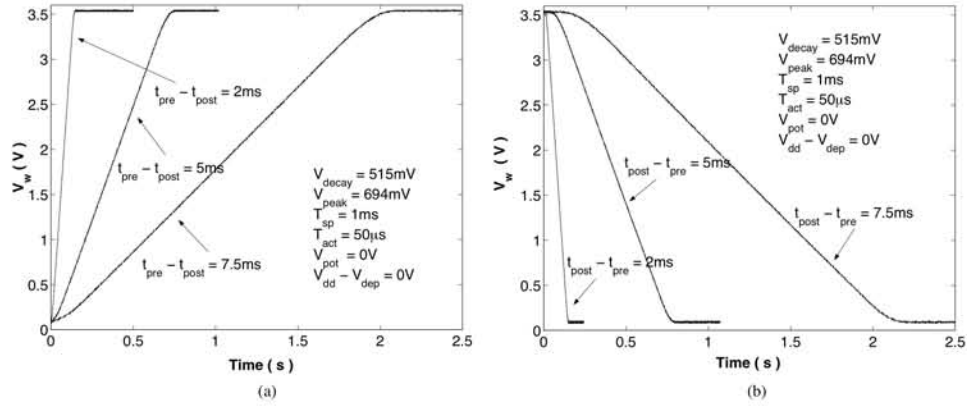

Figure 5: **Test results. Linearity.** (a) The voltage across $C_w$ is initially set to 0V and increased by a sequence of consecutive pairs of pre and postsynaptic spikes. The delays between presynaptic and postsynaptic firing times were set to 2ms, 5ms and 7.5ms (b) The order of pre and postsynaptic spikes is reversed to decrease $V_w$. In both plots the duration of the spikes, $T_{sp}$, and the activation pulse, $T_{act}$, is set to 1ms and and 50$\mu$s respectively.

The learning window plots shown in Figures 6-8 were constructed with test data from a sequence of consecutive presynaptic and postsynaptic spikes with different delays. Before every new pair of presynaptic and postsynaptic spikes, the voltage in $C_w$ was reset to $V_w$=2V. The weight change curves are similar for other initial "reset" weight voltages owing to the linearity of the learning circuit for different $V_w$ values as shown in Figure 5. A power supply voltage of $V_{dd}$=5V is used in all test results shown.

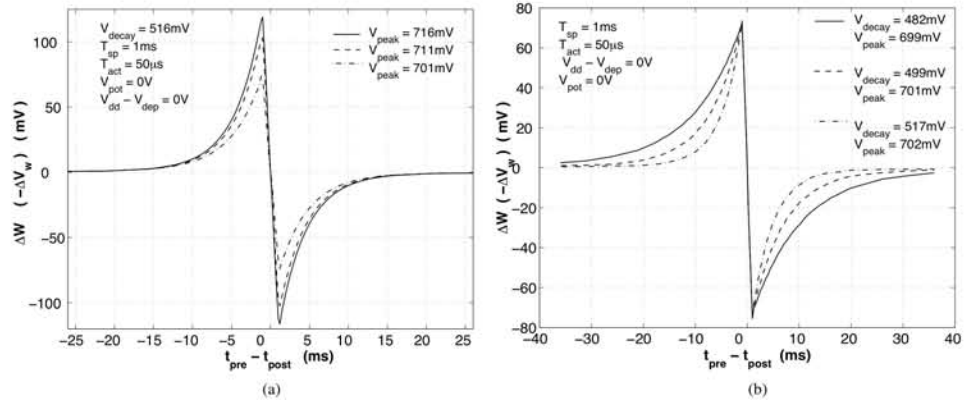

Figure 6: **Test results. (a) Maximum weight change.(b) Learning window decay.** The decay of both tails of the learning window is set by $V_{decay}$. A wide range of time constants can be set. Note, however, that $V_{peak}$ needs to be increased slightly for faster decay rates to maintain exactly the same peak value.

The maximum weight change is easily tuned with $V_{peak}$ as shown in Figure 6(a). Changing the value of $V_{peak}$ modifies by the same amount the absolute value of the peaks at both sides of the curve. The decay of the learning window is controlled by $V_{decay}$. An increase in $V_{decay}$ causes both tails of the learning window to decay faster as seen in Figure 6(b). As mentioned above, matching between both sides of the learning window is possible because the same source of current is used to both increase and decrease the weight.

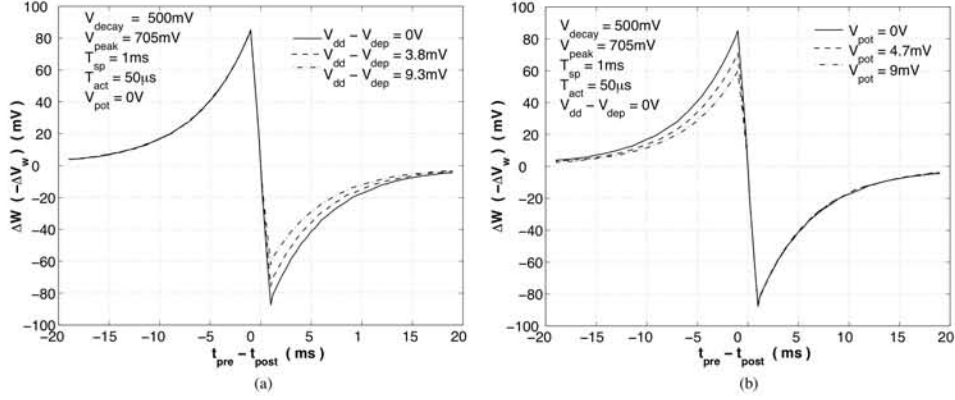

Figure 7: **Test results. Imbalance between potentiation and depression.**

The imbalance between the areas under the potentiation and depression regions of the learning window is a critical parameter of this class of learning rules [3][4]. The circuit proposed can adjust the peak of the curve for potentiation and depression independently (Figure 7). $V_{pot}$ can be used to reduce the area under the potentiation region while keeping unchanged the depression part of the curve, thus setting the overall area under the curve to a negative value (Figure 7(b)). Similarly, with $V_{dd} - V_{dep}$ the area of the depression region can also be reduced (Figure 7(a)).

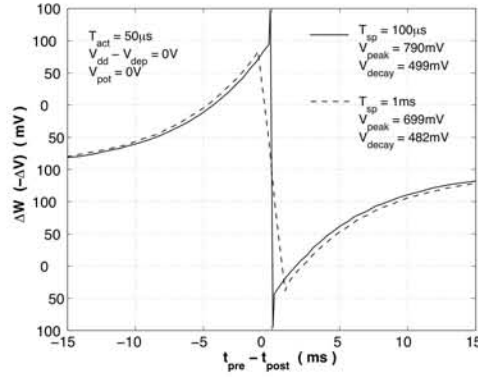

Figure 8: **Test results. Abruptness at the origin**

The abruptness of the learning window at the origin (short delays between pre and postsynaptic spikes) is set by the duration of the spikes. Data in Figure 8 show that the two peaks of the learning window are separated by 2 times the durations of the spikes ($T_{sp}$).

# 4 Discussion and future work

Drawn from experimental data, several temporally-asymmetric Hebbian learning rules have been proposed recently. These learning rules only strengthen the weights when there is a causal relation between presynaptic and postsynaptic activity. Purely random time coincidences between spikes will tend to decrease the weights. Synaptic weight normalization is thus achieved via competition to drive postsynaptic spikes [4]. Predictive sequence learning has been achieved using a similar time-difference learning rule based on the same data [5]. Other pulse-based learning rules have also been used to study how delay tuning could be achieved in the sound source localization system of the barn owl [12].

A simple circuit to implement a general weight change block based on such learning rules has been designed and partially fabricated. The main characteristics of the learning rule, namely the abruptness at the origin, the rate of the decay of the learning window, the imbalance between the potentiation and depression regions and the rate of learning, can be tuned easily. The design also ensures that the circuit can operate at different timescales. As shown, the fabricated circuits have good linearity over a wide range of weight voltage values.

We are currently developing a second chip with a small network of temporally asymmetric Hebbian spiking neurons using the circuit described in this paper. The structure of the network will be reconfigurable. The small network will be used to carry out movement planning experiments by learning of temporal sequences. We envisage the application of networks of temporally-asymmetric Hebbian learning silicon neurons as higher level processing stages for the integration of sensor and motor activities in neuromorphic system. We will concentrate on auditory applications and adaptive, spike-based motion estimation. In both types of application, naturally-occurring correlations in data can be exploited to drive the pulse timing-based learning process.

## Acknowledgements

We thank Robin Woodburn, Patrice Fleury and Martin Reekie for fruitful discussions during the design and tape out of the chip. We also acknowledge that the circuits presented incorporate some of the insights into neuromorphic engineering that one of the authors gained at the Telluride Workshop on Neuromorphic Engineering 2000 (http://www.ini.unizh.ch/telluride2000/).

## References

[1] Markram, H., Lubke, J., Frotscher, M. & Sakmann, B.(1997) Regulation of Synaptic Efficacy by Coincidence of Postsynaptic APs and EPSPs. *Science* **275**, 213-215.

[2] Zhang, L.I., Tao, H.W., Holt, C.E., Harris, W.A. & Poo, M-m.(1998) A critical window for cooperation and competition among developing retinotectal synapses.*Nature* **395**, 37-44.

[3] Abbott, L.F. & Song, S.(1999) Temporally Asymmetric Hebbian Learning, Spike Timing and Neuronal Response Variability. In Kearns, M.S., Solla, S.A., & Cohn, D.A. (eds.), *Advances in Neural Information Processing Systems 11*, 69-75. Cambridge, MA: MIT Press.

[4] Song, S., Miller, K.D. & Abbott, L.F.(2000) Competitive Hebbian Learning Through Spike-Timing Dependent Synaptic Plasticity. *Nature Neuroscience* **3**, 919-926.

[5] Rao, R.P.N., & Sejnowski, T.J.(2000) Predictive Sequence Learning in Recurrent Neocortical Circuits. In Solla, S.A., Leen, T.K. & Muller, K-R. (eds.), *Advances in Neural*

*Information Processing Systems 12*, 164-170. Cambridge, MA: MIT Press.

[6] Murray, A.F. & Smith A.V.W.(1987) Asynchronous Arithmetic for VLSI Neural Systems. *Electronic Letters* **23**, 642-643.

[7] Murray, A.F. & Tarrasenko, L.(1994) *Neural Computing : An Analogue VLSI Approach.* Chapman-Hall.

[8] Hafliger, P., Mahowald, M. & Watts, L. (1996) A Spike Based Learning Neuron in Analog VLSI. In Mozer, M.C., Jordan, M.I., & Petsche, T. (eds.), *Advances in Neural Information Processing Systems 9*, 692-698. Cambridge, MA: MIT Press.

[9] Indiveri, G.(2000) Modeling Selective Attention Using a Neuromorphic Analog VLSI Device. *Neural computation* **12**, 2857-2880.

[10] van Schaik, A., Fragniére, E. & Vittoz, E.(1996) An Analogue Electronic Model of Ventral Cochlear Nucleus Neurons. In *Proceedings of the 5th International Conference on Microelectronics for Neural, Fuzzy and Bio-inspired Systems; Microneuro '96*, 52-59. Los Alamitos, CA: IEEE Computer Society Press.

[11] Shams, M., Ebergen, J.C. & Elmasry, M.I. (1998) Modeling and Comparing CMOS Implementations of the C-elment. IEEE Transactions on Very Large Scale Intergration (VLSI) Systems, Vol. 6, No. 4, 563-567.

[12] Gerstner, W., Kempter, R., van Hemmen, J.L. & Wagner, H.(1999) Hebbian Learning of Pulse Timing in the Barn Owl Auditory System. In Mass, W. & Bishop, C.M. (eds.), *Pulsed Neural Networks.* Cambridge, MA: MIT Press.
